# Meiosis Networks

**Stephen José Hanson** [1]
Learning and Knowledge Acquisition Group
Siemens Research Center
Princeton, NJ  08540

## ABSTRACT

A central problem in connectionist modelling is the control of network and architectural resources during learning. In the present approach, weights reflect a coarse prediction history as coded by a distribution of values and parameterized in the mean and standard deviation of these weight distributions. Weight updates are a function of both the mean and standard deviation of each connection in the network and vary as a function of the error signal ("stochastic delta rule"; Hanson, 1990). Consequently, the weights maintain information on their central tendency and their "uncertainty" in prediction. Such information is useful in establishing a policy concerning the size of the nodal complexity of the network and growth of new nodes. For example, during problem solving the present network can undergo "meiosis", producing two nodes where there was one "overtaxed" node as measured by its coefficient of variation. It is shown in a number of benchmark problems that meiosis networks can find minimal architectures, reduce computational complexity, and overall increase the efficiency of the representation learning interaction.

[1] Also a member of the Cognitive Science Laboratory, Princeton University, Princeton, NJ 08542

# 1  INTRODUCTION

Search problems which involve high dimensionality, a-priori constraints and nonlinearities are hard. Unfortunately, learning problems in biological systems involve just these sorts of properties. Worse, one can characterize the sort of problem that organisms probably encounter in the real world as those that do not easily admit solutions that involve, simple averaging, optimality, linear approximation or complete knowledge of data or nature of the problem being solved. We would contend there are three basic properties of real learning result in an ill-defined set problems and heterogeneous set of solutions:

- Data are continuously available but incomplete; the learner must constantly update parameter estimates with stingy bits of data which may represent a very small sample from the possible population

- Conditional distributions of response categories with respect to given features are unknown and must be estimated from possibly unrepresentative samples.

- Local (in time) information may be misleading, wrong, or nonstationary, consequently there is a poor tradeoff between the present use of data and waiting for more and possibly flawed data-- consequently updates must be small and revocable.

These sorts of properties represent only one aspect of the learning problem faced by real organisms in real environments. Nonetheless, they underscore why "weak" methods--methods that assume little about the environment in which they are operating --are so critical.

## 1.1  LEARNING AND SEARCH

It is possible to precisely characterize the search problem in terms of the resources or degress of freedom in the learning model. If the task the learning system is to perform is classification then the system can be analyzed in terms of its ability to dichotomize stimulus points in feature space.

*Dichotomization Capability: Network Capacity* Using a linear *fan-in* or hyperplane type neuron we can characterize the degrees of freedom inherent in a network of units with thresholded output. For example, with linear boundaries, consider 4 points, well distributed in a 2-dimensional feature space. There are exactly 14 linearly separable dichotomies that can be formed with the 4 target points. However, there are actually 16 ($2^4$) possible dichotomies of 4 points in 2 dimensions consequently, the number of possible dichotomies or arbitrary categories that are linearly implementable can be thought of as a capacity of the linear network in $k$ dimensions with $n$ examples. The general category capacity measure (Cover, 1965) can be written as:

$$C(n,k)=2\sum_{j=0}^{k}\frac{(n-1)!}{(n-1-j)!\,j!},\quad n>k+1 \tag{1}$$

Note the dramatic growth in C as a function of k, the number of feature dimensions, for example, for 25 stimuli in a 5 dimensional feature space there are 100,670 linear dichotomies. Undertermination in these sorts of linear networks is the rule not the exception. This makes the search process and the nature of constraints on the search process critical in finding solutions that may be useful in the given problem domain.

## 1.2    THE STOCHASTIC DELTA RULE

Actual mammalian neural systems involve noise. Responses from the same individual unit in isolated cortex due to cyclically repeated identical stimuli will never result in identical bursts Transmission of excitation through neural networks in living systems is essentially stochastic in nature. The typical activation function used in connectionist models must be assumed to be an average over many intervals, since any particular neuronal pulse train appears quite random [in fact, Poisson; for example see Burns,1968; Tomko & Crapper, 1974].

This suggests that a particular neural signal in time may be modeled by a *distribution* of synaptic values rather then a single value. Further this sort of representation provides a natural way to affect the synaptic efficacy in time. In order to introduce noise adaptively, we require that the synaptic modification be a function of a random increment or decrement proportional in size to the present error signal. Consequently, the weight delta or gradient itself becomes a random variable based on prediction performance. Thus, the noise that seems ubiquitous and apparently useless throughout the nervous system can be turned to at least three advantages in that it provides the system with mechanisms for (1) entertaining multiple response hypotheses given a single input (2) maintaining a coarse prediction history that is local, recent, and cheap, thus providing punctate credit assignment opportunities and finally, (3) revoking parameterizations that are easy to reach, locally stable, but distant from a solution.

Although it is possible to implement the present principle a number of different ways we chose to consider a connection strength to be represented as a distribution of weights with a finite mean and variance (see Figure 1).

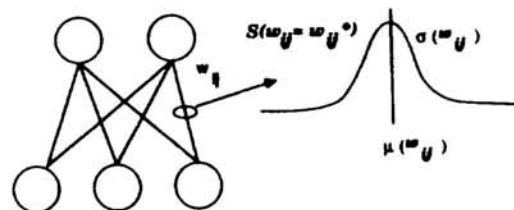

**Figure 1:** Weights as Sampling Distributions

A forward activation or recognition pass consists of randomly sampling a weight from the existing distribution calculating the dot product and producing an output

for that pass.

$$x_i = \sum_j w_{ij}^* y_j \qquad (2)$$

where the sample is found from,

$$S(w_{ij}=w_{ij}^*) = \mu_{w_{ij}} + \sigma_{w_{ij}} \phi(w_{ij};0,1) \qquad (3)$$

Consequently $S(w_{ij}=w_{ij}^*)$ is a random variable constructed from a finite mean $\mu_{w_{ij}}$ and standard deviation $\sigma_{w_{ij}}$ based on a normal random variate ($\phi$) with mean zero and standard deviation one. Forward recognition passes are therefore one to many mappings, each sampling producing a different weight depending on the mean and standard deviation of the particular connection while the system remains stochastic.

In the present implementation there are actually three separate equations for learning. The mean of the weight distribution is modified as a function of the usual gradient based upon the error, however, note that the random sample point is retained for this gradient calculation and is used to update the mean of the distribution for that synapse.

$$\mu_{w_{ij}}(n+1) = \alpha \left(-\frac{\partial E}{\partial w_{ij}^*}\right) + \mu_{w_{ij}}(n) \qquad (4)$$

Similarly the standard deviation of the weight distribution is modified as a function of the gradient, however, the sign of the gradient is ignored and the update can only increase the variance if an error results. Thus errors immediately increase the variance of the synapse to which they may be attributed.

$$\sigma_{w_{ij}}(n+1) = \beta \left| -\frac{\partial E}{\partial w_{ij}^*} \right| + \sigma_{w_{ij}}(n) \qquad (5)$$

A third and final learning rule determines the decay of the variance of synapses in the network,

$$\sigma_{w_{ij}}(n+1) = \varsigma\sigma_{w_{ij}}(n), \quad \varsigma<1. \qquad (6)$$

As the system evolves for $\varsigma$ less than one, the last equation of this set guarantees that the variances of all synapses approach zero and that the system itself becomes deterministic prior to solution. For small $\varsigma$ the system evolves very rapidly to deterministic, while larger $\varsigma$s allow the system to revisit chaotic states as needed during convergence. A simpler implementation of this algorithm involves just the gradient itself as a random variable (hence, the name "stochastic delta rule"), however this approach confounds the growth in variance of the weight distribution with the decay and makes parametric studies more complicated to implement.

The stochastic delta rule implements a local, adaptive simulated annealing (cf. Kirkpatrick, S., Gelatt, C. D. & Veechi, M., 1983) process occuring at different rates in the network dependent on prediction history. Various benchmark tests of this

basic algorithm are discussed in Hanson (1990).

## 1.3    MEIOSIS

In the SDR rule disscussed above, the standard deviation of the weight distributions might be seen as uncertainty measure concerning the weight value and strength. Consequently, changes in the standard deviation can be taken as a measure of the "prediction value" of the connection. Hidden units with significant uncertainty have low prediction value and are performing poorly in reducing errors. If hidden unit uncertainty increases beyond the cumulative weight value or "signal" to that unit then the complexity of the architecture can be traded off with the uncertainty per unit. Consequently, the unit "splits" into two units each copying half the architecture information to each of the new two units.

Networks are initialized with a random mean and variance values (where the variance is started in the interval (10,-10)). Number of hidden units in all problems was initialized at one. The splitting policy is fixed for all problems to occur when both the C.V. (standard deviation relative to the mean) for the input and output to the hidden unit exceeds 100%, that is, when the composite variance of the connection strengths is 100% of the composite mean value of the connection strengths:

$$\frac{\sum_i \sigma_{ij}}{\sum_i \mu_{ij}} > 1.0 \text{ and } \frac{\sum_k \sigma_{jk}}{\sum_k \mu_{jk}} > 1.0$$

Meiosis then proceeds as follows (see Figure 2)

- A forward stochastic pass is made producing an output
- Output is compared to target producing errors which are then used to update the mean and variance of weight.
- The composite input and output variance and means are computed for each hidden units
- For those hidden units whose composite C.V.s are > 1.0 node splitting occurs; half the variance is assigned to each new node with a jittered mean centered at the old mean

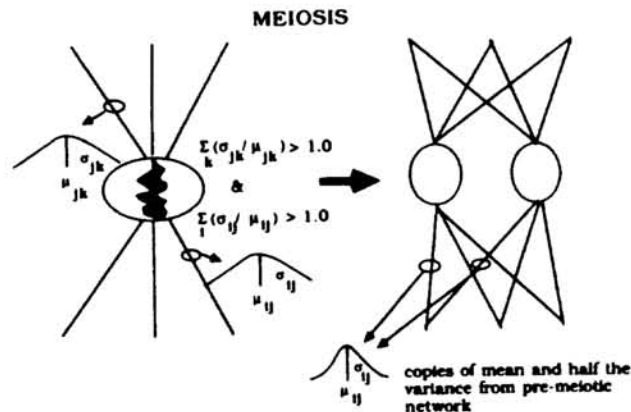

**Figure 2:** Meiosis

There is no stopping criteria. The network stops creating nodes based on the prediction error and noise level ( $\beta,\varsigma$ ) .

## 1.4   EXAMPLES

### 1.4.1   Parity Benchmark: Finding the Right number of units

Small parity problems (Exclusive-or and 3BIT parity) were used to explore sensitivity of the noise parameters on node splitting and to benchmark the method. All runs were with fixed learning rate ( $\eta = .5$ ) and momentum ( $\alpha = .75$). Low values of zeta ($<.7$) produce minimal or no node splitting, while higher values ($>.99$) seem to produce continuous node spliting without regard to the problem type. Zeta was fixed (.98) and beta, the noise per step parameter was varied between values .1 and .5. The following runs were unaffected by varying beta between these two values.

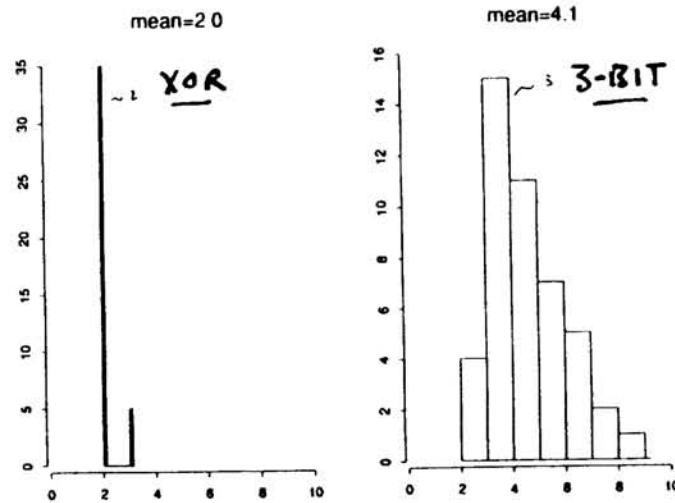

**Figure 3:** Number of Hidden Units at Convergence

Shown in Figure 3 are 50 runs of Exclusive-or and 50 runs of 3 BIT PARITY. Histograms show for exclusive-or that almost all runs ($>95\%$) ended up with 2 hidden units while for the 3BIT PARITY case most runs produce 3 hidden units, however with considerably more variance, some ending with 2 while a few runs ended with as many 9 hidden units. The next figure (Figure 4) shows histograms for

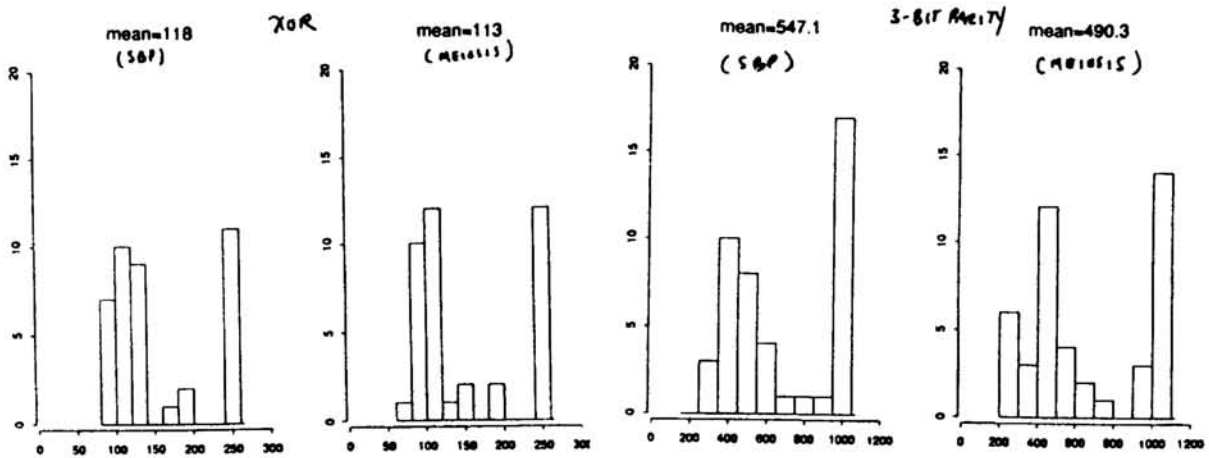

**Figure 4:** Convergence Times

the convergence time showing a slight advantage in terms of convergence for the meiosis networks for both exclusive-or and 3 BIT PARITY.

### 1.4.2    Blood NMR Data: Nonlinear Separability

In the Figure 5 data were taken from 10 different continuous kinds of blood measurements, including, total lipid content, cholesterol (mg/dl), High density lipids, low-density lipids, triglyceride, etc as well as some NMR measures. Subjects were previously diagnosed for presence (C) or absence (N) of a blood disease.

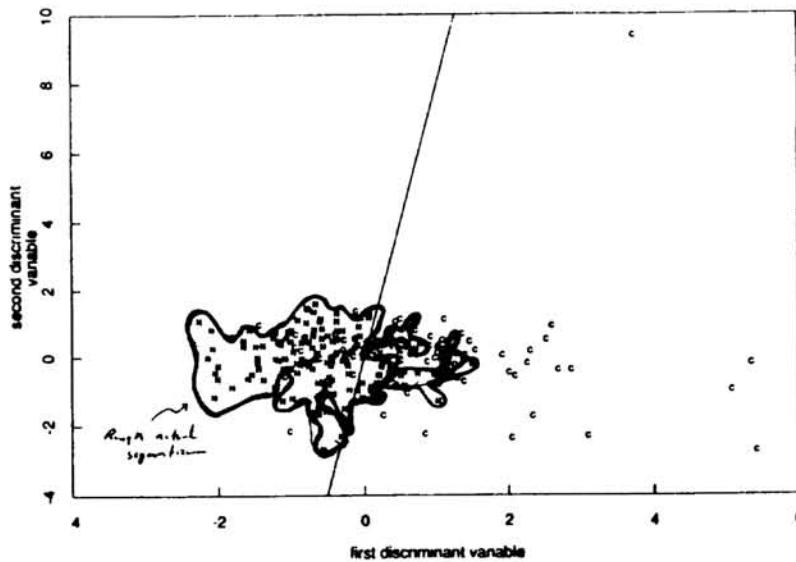

**Figure 5:** Blood NMR Separability

The data consisted of 238 samples, 146 Ns and 92 Cs. Shown in the adjoining figure is a Perceptron (linear discriminant analysis) response to the data. Each original data point is projected into the first two discriminant variables showing about 75% of the data to be linearly separable (k-k/2 jackknife tests indicate about 52% transfer rates). However, also shown is a rough non-linear envelope around one class of

subjects(N) showing the potentially complex decision region for this data.

### 1.4.3    Meiosis Learning curves

Data was split into two groups (118,120) for learning and transfer tests. Learning curves for both the meiosis network and standard back-propagation are shown in the Figure 6. Also shown in this display is the splitting rate for the meiosis network showing it grow to 7 hidden units and freezing during the first 20 sweeps.

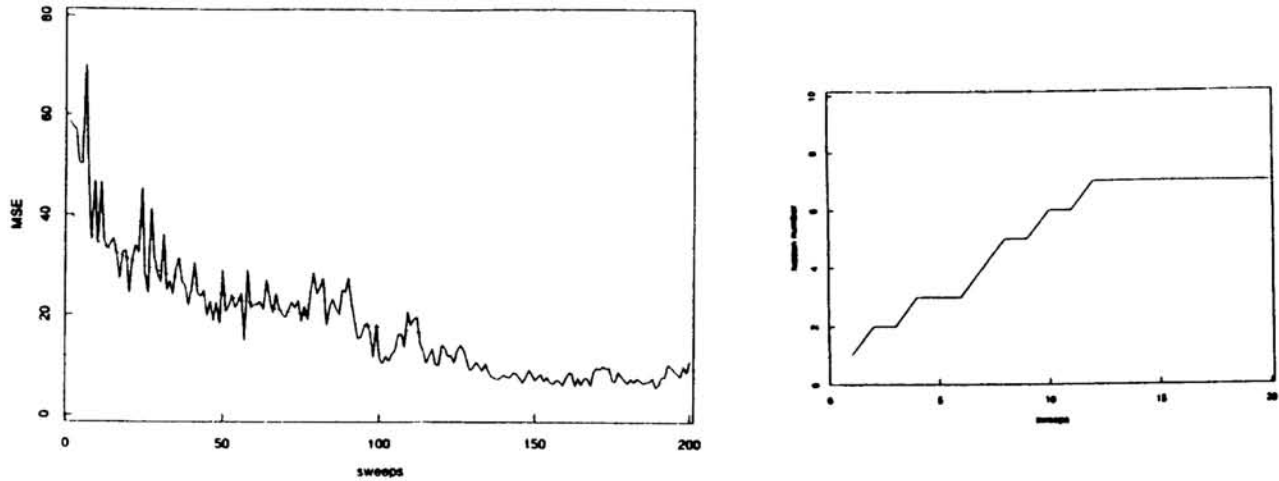

**Figure 6:** Learning Curves and Splitting Rate

### 1.4.4    Transfer Rate

Backpropagation was run on the blood data with 0 (perceptron), 2, 3, 4, 5, 6, 7, and 20 hidden units. Shown is the median transfer rate of 3 runs for each hidden unit network size. Transfer rate seemed to hover near 65% as the number of hidden units approached 20. A meiosis network was also run 3 times on the data (using $\beta$ .40 and $\varsigma$ .98). Transfer Rate shown in Figure 7 was always above 70% at the 7 hidden unit number.

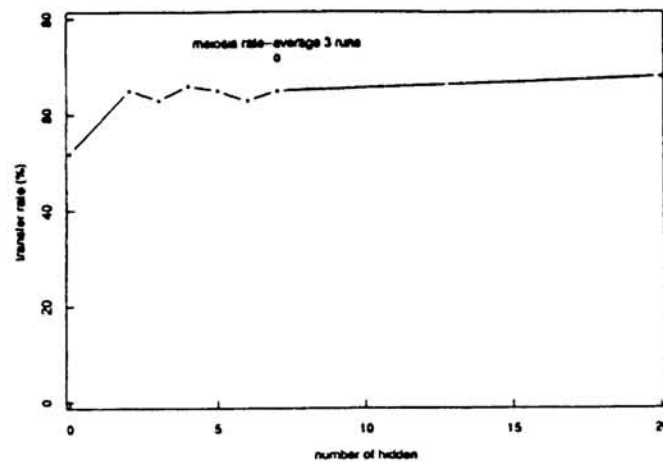

**Figure 7:** Transfer Rate as a Function of Hidden Unit Number

## 1.5   Conclusions

The key property of the present scheme is the integration of representational aspects that are sensitive to network prediction and at the same time control the architectural resources of the network. Consequently, with Meiosis networks it is possible to dynamically and opportunistically control network complexity and therefore indirectly its learning efficiency and generalization capacity. Meiosis Networks were defined upon earlier work using local noise injections and noise related learning rules. As learning proceeds the meiosis network can measure the prediction history of particular nodes and if found to be poor, can split the node and opportunistically to increase the resources of the network. Further experiments are required in order to understand different advantages of splitting policies and their affects on generalization and speed of learning.

## References

Burns, B. D  The uncertain nervous system, London· Edward Arnold Ltd , 1968.

Cover, T. M. Geometrical and statistical properties of systems of linear inequalities with applications to pattern recognition. IEEE Trans Elec Computers, Vol EC-14,3, pp 236-334, 1965

Hanson, S. J. A stochastic version of the delta rule  Physica D, 1990.

Hanson, S  J  & Burr D  J  Minkowski Back-propagation. learning in connectionist models with non-euclidean error signals, Neural Information Processing Systems, American Institute of Physics 1988

Hanson, S  J  & Pratt, L.  A comparison of different biases for minimal network construction with back-propagation, Advances in Neural Information Processing, D. Touretzsky, Morgan-Kaufmann, 1989

Kirkpatrick, S , Gelatt, C  D.  & Veechi, M.  Optimization by simulated annealing, Science, 220, 671-680, 1983.

Tomko, G. J. & Crapper, D. R  Neural variability  Non-stationary response to identical visual stimuli, Brain Research, 79, p. 405-418, 1974
